# Efficient Kernel Machines Using the Improved Fast Gauss Transform

**Changjiang Yang, Ramani Duraiswami and Larry Davis**
Department of Computer Science, Perceptual Interfaces and Reality Laboratory
University of Maryland, College Park, MD 20742
{yangcj,ramani,lsd}@umiacs.umd.edu

## Abstract

The computation and memory required for kernel machines with $N$ training samples is at least $O(N^2)$. Such a complexity is significant even for moderate size problems and is prohibitive for large datasets. We present an approximation technique based on the improved fast Gauss transform to reduce the computation to $O(N)$. We also give an error bound for the approximation, and provide experimental results on the UCI datasets.

## 1 Introduction

Kernel based methods, including support vector machines [16], regularization networks [5] and Gaussian processes [18], have attracted much attention in machine learning. The solid theoretical foundations and good practical performance of kernel methods make them very popular. However one major drawback of the kernel methods is their scalability. Kernel methods require $O(N^2)$ storage and $O(N^3)$ operations for direct methods, or $O(N^2)$ operations per iteration for iterative methods, which is impractical for large datasets.

To deal with this scalability problem, many approaches have been proposed, including the Nyström method [19], sparse greedy approximation [13, 12], low rank kernel approximation [3] and reduced support vector machines [9]. All these try to find a reduced subset of the original dataset using either random selection or greedy approximation. In these methods there is no guarantee on the approximation of the kernel matrix in a deterministic sense. An assumption made in these methods is that most eigenvalues of the kernel matrix are zero. This is not always true and its violation results in either performance degradation or negligible reduction in computational time or memory.

We explore a deterministic method to speed up kernel machines using the improved fast Gauss transform (IFGT) [20, 21]. The kernel machine is solved iteratively using the conjugate gradient method, where the dominant computation is the matrix-vector product which we accelerate using the IFGT. Rather than approximating the kernel matrix by a low-rank representation, we approximate the matrix-vector product by the improved fast Gauss transform to any desired precision. The total computational and storage costs are of linear order in the size of the dataset. We present the application of the IFGT to kernel methods in the context of the Regularized Least-Squares Classification (RLSC) [11, 10], though the approach is general and can be extended to other kernel methods.

## 2 Regularized Least-Squares Classification

The RLSC algorithm [11, 10] solves the binary classification problems in Reproducing Kernel Hilbert Space (RKHS) [17]: given $N$ training samples in $d$-dimensional space $\mathbf{x}_i \in$

$\mathcal{R}^d$ and the labels $y_i \in \{-1, 1\}$, find $f \in \mathcal{H}$ that minimizes the regularized risk functional

$$\min_{f \in \mathcal{H}} \frac{1}{N} \sum_{i=1}^{N} V(y_i, f(\mathbf{x}_i)) + \lambda \|f\|_K^2, \tag{1}$$

where $\mathcal{H}$ is an RKHS with reproducing kernel $K$, $V$ is a convex cost function and $\lambda$ is the regularization parameter controlling the tradeoff between the cost and the smoothness. Based on the Representer Theorem [17], the solution has a representation as

$$f_\lambda(\mathbf{x}) = \sum_{i=1}^{N} c_i K(\mathbf{x}, \mathbf{x}_i). \tag{2}$$

If the loss function $V$ is the hinge function, $V(y, f) = (1 - yf)_+$, where $(\tau)_+ = \tau$ for $\tau > 0$ and 0 otherwise, then the minimization of (1) leads to the popular Support Vector Machines which can be solved using quadratic programming.

If the loss function $V$ is the square-loss function, $V(y, f) = (y - f)^2$, the minimization of (1) leads to the so-called Regularized Least-Squares Classification which requires only the solution of a linear system. The algorithm has been rediscovered several times and has many different names [11, 10, 4, 15]. In this paper, we stick to the term "RLSC" for consistency. It has been shown in [11, 4] that RLSC achieves accuracy comparable to the popular SVMs for binary classification problems.

If we substitute (2) into (1), and denote $\mathbf{c} = [c_1, \ldots, c_N]^T$, $K = K(\mathbf{x}_i, \mathbf{x}_j)$, we can find the solution of (1) by solving the linear system

$$(K + \lambda' I)\mathbf{c} = \mathbf{y} \tag{3}$$

where $\lambda' = \lambda N$, $I$ is the identity matrix, and $\mathbf{y} = [y_1, \ldots, y_N]^T$.

There are many choices for the kernel function $K$. The Gaussian is a good kernel for classification and is used in many applications. If a Gaussian kernel is applied, as shown in [10], the classification problem can be solved by the solution of a linear system, i.e., Regularized Least-Squares Classification. A direct solution of the linear system will require $O(N^3)$ computation and $O(N^2)$ storage, which is impractical even for problems of moderate size.

---

**Algorithm 1** Regularized Least-Squares Classification

---

**Require:** Training dataset $S_N = (\mathbf{x}_i, y_i)_{i=1}^{N}$.
   1. Choose the Gaussian kernel: $K(\mathbf{x}, \mathbf{x}') = e^{-\|\mathbf{x} - \mathbf{x}'\|^2/\sigma^2}$.
   2. Find the solution as $f(\mathbf{x}) = \sum_{i=1}^{N} c_i K(\mathbf{x}, \mathbf{x}_i)$, where $\mathbf{c}$ satisfies the linear system (3).
   3. Solve the linear system (3).

---

An effective way to solve the large-scale linear system (3) is to use iterative methods. Since the matrix $K$ is symmetric, we consider the well-known *conjugate gradient* method. The conjugate gradient method solves the linear system (3) by iteratively performing the matrix-vector multiplication $K\mathbf{c}$. If $rank(K) = r$, then the conjugate gradient algorithm converges in at most $r + 1$ steps. Only one matrix-vector multiplication and $10N$ arithmetic operations are required per iteration. Only four $N$-vectors are required for storage. So the computational complexity is $O(N^2)$ for low-rank $K$ and the storage requirement is $O(N^2)$. While this represents an improvement for most problems, the rank of the matrix may not be small, and moreover the quadratic storage and computational complexity are still too high for large datasets. In the following sections, we present an algorithm to reduce the computational and storage complexity to linear order.

# 3 Fast Gauss Transform

The matrix-vector product $K\mathbf{c}$ can be written in the form of the so-called *discrete Gauss transform* [8]

$$G(\mathbf{y}_j) = \sum_{i=1}^{N} c_i e^{-\|\mathbf{x}_i - \mathbf{y}_j\|^2/\sigma^2}, \tag{4}$$

where $c_i$ are the weight coefficients, $\{\mathbf{x}_i\}_{i=1}^{N}$ are the centers of the Gaussians (called "sources"), and $\sigma$ is the bandwidth parameter of the Gaussians. The sum of the Gaussians is evaluated at each of the "target" points $\{\mathbf{y}_j\}_{j=1}^{M}$. Direct evaluation of the Gauss transform at $M$ target points due to $N$ sources requires $O(MN)$ operations.

The Fast Gauss Transform (FGT) was invented by Greengard and Strain [8] for efficient evaluation of the Gauss transform in $O(M + N)$ operations. It is an important variant of the more general Fast Multipole Method [7].

The FGT [8] expands the Gaussian function into Hermite functions. The expansion of the univariate Gaussian is

$$e^{-\|y_j - x_i\|^2/\sigma^2} = \sum_{n=0}^{p-1} \frac{1}{n!} \left(\frac{x_i - x_*}{\sigma}\right)^n h_n \left(\frac{y_j - x_*}{\sigma}\right) + \epsilon(p), \tag{5}$$

where $h_n(x)$ are the Hermite functions defined by $h_n(x) = (-1)^n \frac{d^n}{dx^n}\left(e^{-x^2}\right)$, and $x_*$ is the expansion center. The $d$-dimensional Gaussian function is treated as a Kronecker product of $d$ univariate Gaussians. For simplicity, we adopt the multi-index notation of the original FGT papers [8]. A multi-index $\alpha = (\alpha_1, \ldots, \alpha_d)$ is a $d$-tuple of nonnegative integers. For any multi-index $\alpha \in \mathbf{N}^d$ and any $x \in \mathbf{R}^d$, we have the monomial $x^\alpha = x_1^{\alpha_1} x_2^{\alpha_2} \cdots x_d^{\alpha_d}$. The length and the factorial of $\alpha$ are defined as $|\alpha| = \alpha_1 + \alpha_2 + \ldots + \alpha_d$, $\alpha! = \alpha_1! \alpha_2! \cdots \alpha_d!$. The multidimensional Hermite functions are defined by

$$h_\alpha(x) = h_{\alpha_1}(x_1) h_{\alpha_2}(x_2) \cdots h_{\alpha_d}(x_d).$$

The sum (4) is then equal to the Hermite expansion about center $x_*$:

$$G(y_j) = \sum_{\alpha \geq 0} C_\alpha h_\alpha \left(\frac{y_j - x_*}{h}\right), \quad C_\alpha = \frac{1}{\alpha!} \sum_{i=1}^{N} c_i \left(\frac{x_i - x_*}{h}\right)^\alpha. \tag{6}$$

where $C_\alpha$ are the coefficients of the Hermite expansions.

If we truncate each of the Hermite series (6) after $p$ terms (or equivalently order $p-1$), then each of the coefficients $C_\alpha$ is a $d$-dimensional matrix with $p^d$ terms. The total computational complexity for a single Hermite expansion is $O((M + N)p^d)$. The factor $O(p^d)$ **grows exponentially** as the dimensionality $d$ increases. Despite this defect in higher dimensions, the FGT is quite effective for two and three-dimensional problems, and has achieved success in some physics, computer vision and pattern recognition applications.

In practice a single expansion about one center is not always valid or accurate over the entire domain. A space subdivision scheme is applied in the FGT and the Gaussian functions are expanded at multiple centers. The original FGT subdivides space into uniform boxes, which is simple, but highly inefficient in higher dimensions. The number of boxes grows exponentially with dimensionality, which makes it inefficient for storage and for searching nonempty neighbor boxes. Most important, since the ratio of volume of the hypercube to that of the inscribed sphere grows exponentially with dimension, points have a high probability of falling into the area inside the box and outside the sphere, where the truncation error of the Hermite expansion is much larger than inside of the sphere.

### 3.1 Improved Fast Gauss Transform

In brief, the original FGT suffers from the following two defects:

1. The exponential growth of computationally complexity with dimensionality.
2. The use of the box data structure in the FGT is inefficient in higher dimensions.

We introduced the improved FGT [20, 21] to address these deficiencies, and it is summarized below.

#### 3.1.1 Multivariate Taylor Expansions

Instead of expanding the Gaussian into Hermite functions, we factorize it as

$$e^{-\|y_j - x_i\|^2/\sigma^2} = e^{-\|\Delta y_j\|^2/\sigma^2} \, e^{-\|\Delta x_i\|^2/\sigma^2} \, e^{2\Delta y_j \cdot \Delta x_i/\sigma^2}, \qquad (7)$$

where $x_*$ is the center of the sources, $\Delta y_j = y_j - x_*, \Delta x_i = x_i - x_*$. The first two exponential terms can be evaluated individually at the source points or target points. In the third term, the sources and the targets are entangled. Here we break the entanglement by expanding it into a multivariate Taylor series

$$e^{2\Delta y_j \cdot \Delta x_i/\sigma^2} = \sum_{n=0}^{\infty} 2^n \left( \frac{\Delta x_i}{\sigma} \cdot \frac{\Delta y_j}{\sigma} \right)^n = \sum_{|\alpha| \geq 0} \frac{2^{|\alpha|}}{\alpha!} \left( \frac{\Delta x_i}{\sigma} \right)^\alpha \left( \frac{\Delta y_j}{\sigma} \right)^\alpha. \qquad (8)$$

If we truncate the series after total order $p - 1$, then the number of terms is $r_{p-1,d} = \binom{p+d-1}{d}$ which is much less than $p^d$ in higher dimensions. For $d = 12$ and $p = 10$, the original FGT needs $10^{12}$ terms, while the multivariate Taylor expansion needs only 293930. For $d \to \infty$ and moderate $p$, the number of terms is $O(d^p)$, a substantial reduction.

From Eqs.(7) and (8), the weighted sum of Gaussians (4) can be expressed as a multivariate Taylor expansions about center $x_*$:

$$G(y_j) = \sum_{|\alpha| \geq 0} C_\alpha e^{-\|y_j - x_*\|^2/\sigma^2} \left( \frac{y_j - x_*}{\sigma} \right)^\alpha, \qquad (9)$$

where the coefficients $C_\alpha$ are given by

$$C_\alpha = \frac{2^{|\alpha|}}{\alpha!} \sum_{i=1}^{N} c_i e^{-\|x_i - x_*\|^2/\sigma^2} \left( \frac{x_i - x_*}{\sigma} \right)^\alpha. \qquad (10)$$

The coefficients $C_\alpha$ can be efficiently evaluate with $r_{nd}$ storage and $r_{nd} - 1$ multiplications using the multivariate Horner's rule [20].

#### 3.1.2 Spatial Data Structures

To efficiently subdivide the space, we need a scheme that adaptively subdivides the space according to the distribution of points. It is also desirable to generate cells as compact as possible. Based on these consideration, we model the space subdivision task as a $k$-center problem [1]: given a set of $N$ points and a predefined number of clusters $k$, find a partition of the points into clusters $S_1, \ldots, S_k$, with cluster centers $c_1, \ldots, c_k$, that minimizes the maximum radius of any cluster:

$$\max_i \max_{v \in S_i} \|v - c_i\|.$$

The $k$-center problem is known to be $NP$-hard. Gonzalez [6] proposed a very simple greedy algorithm, called *farthest-point clustering*. Initially, pick an arbitrary point $v_0$ as the center of the first cluster and add it to the center set $C$. Then, for $i = 1$ to $k$ do the follows: in iteration $i$, for every point, compute its distance to the set $C$: $d_i(v, C) = \min_{c \in C} \|v - c\|$. Let $v_i$ be a point that is farthest away from $C$, i.e., a point for which $d_i(v_i, C) = \max_v d_i(v, C)$. Add $v_i$ to the center set $C$. After $k$ iterations, report the points $v_0, v_1, \ldots, v_{k-1}$ as the cluster centers. Each point is then assigned to its nearest center.

Gonzalez [6] proved that farthest-point clustering is a 2-approximation algorithm, i.e., it computes a partition with maximum radius at most twice the optimum. The direct implementation of farthest-point clustering has running time $O(Nk)$. Feder and Greene [2] give a two-phase algorithm with optimal running time $O(N \log k)$. In practice, we used circular lists to index the points and achieve the complexity $O(N \log k)$ empirically.

### 3.1.3 The Algorithm and Error Bound

The improved fast Gauss transform consists of the following steps:

---

**Algorithm 2** Improved Fast Gauss Transform

---

1. Assign $N$ sources into $k$ clusters using the farthest-point clustering algorithm such that the radius is less than $\sigma \rho_x$.
2. Choose $p$ sufficiently large such that the error estimate (11) is less than the desired precision $\epsilon$.
3. For each cluster $S_k$ with center $c_k$, compute the coefficients given by (10).
4. Repeat for each target $y_j$, find its neighbor clusters whose centers lie within the range $\sigma \rho_y$. Then the sum of Gaussians (4) can be evaluated by the expression (9).

---

The amount of work required in step 1 is $O(N \log k)$ using Feder and Greene's algorithm [2]. The amount of work required in step 3 is of $O(N\ r_{pd})$. The work required in step 4 is $O(Mn\ r_{pd})$, where $n \leq k$ is the maximum number of neighbor clusters for each target. So, the improved fast Gauss transform achieves linear running time. The algorithm needs to store the $k$ coefficients of size $r_{pd}$, so the storage complexity is reduced to $O(Kr_{pd})$. To verify the linear order of our algorithm, we generate $N$ source points and $N$ target points in $4, 6, 8, 10$ dimensional unit hypercubes using a uniform distribution. The weights on the source points are generated from a uniform distribution in the interval $[0, 1]$ and $\sigma = 1$. The results of the IFGT and the direct evaluation are displayed in Figure 1(a), (b), and confirm the linear order of the IFGT.

The error of the improved fast Gauss transform (2) is bounded by

$$|E(G(y_j))| \leq \sum_{i=1}^{N} |c_i| \left( \frac{2^p}{p!} \rho_x^p \rho_y^p + e^{-(\rho_y - \rho_x)^2} \right). \tag{11}$$

The details are in [21]. The comparison between the maximum absolute errors in the simulation and the estimated error bound (11) is displayed in Figure 1(c) and (d). It shows that the error bound is very conservative compared with the real errors. Empirically we can obtain the parameters on a randomly selected subset and use them on the entire dataset.

## 4 IFGT Accelerated RLSC: Discussion and Experiments

The key idea of all acceleration methods is to reduce the cost of the matrix-vector product. In reduced subset methods, this is performed by evaluating the product at a few points, assuming that the matrix is low rank. The general Fast Multipole Methods (FMM) seek to analytically approximate the possibly full-rank matrix as a *sum* of low rank approximations with a tight error bound [14] (The FGT is a variant of the FMM with Gaussian kernel). It is expected that these methods can be more robust, while at the same time achieve significant acceleration.

The problems to which kernel methods are usually applied are in higher dimensions, though the intrinsic dimensionality of the data is expected to be much smaller. The original FGT does not scale well to higher dimensions. Its cost is of linear order in the number of samples, but exponential order in the number of dimensions. The improved FGT uses new data structures and a modified expansion to reduce this to polynomial order.

Despite this improvement, at first glance, even with the use of the IFGT, it is not clear if the reduction in complexity will be competitive with the other approaches proposed. Reason

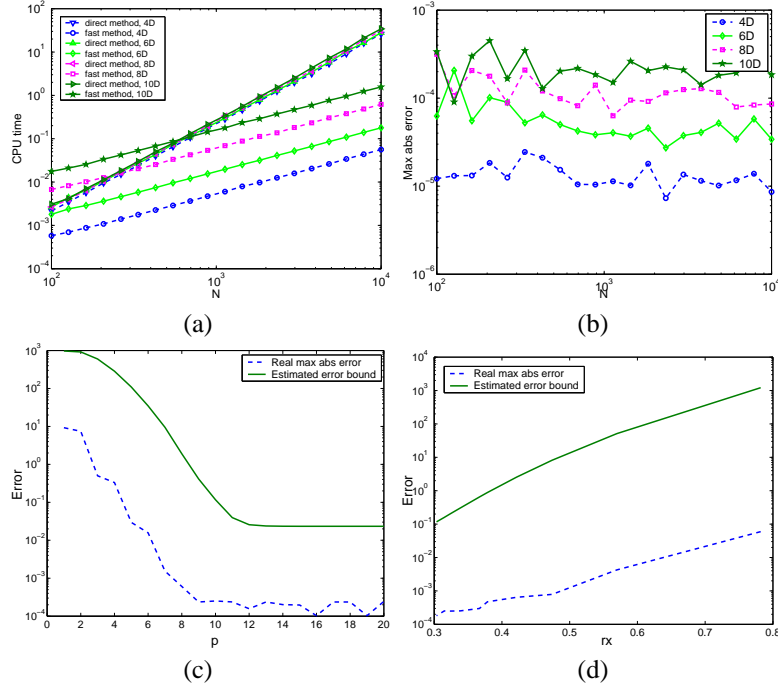

**Figure 1:** (a) Running time and (b) maximum absolute error *w.r.t.* $N$ in $d = 4, 6, 8, 10$. The comparison between the real maximum absolute errors and the estimated error bound (11) *w.r.t.* (c) the order of the Taylor series $p$, and (d) the radius of the farthest-point clustering algorithm $r_x = \sigma \rho_x$. The uniformly distributed sources and target points are in 4-dimension.

for hope is provided by the fact that in high dimensions we expect that the IFGT with very low order expansions will converge rapidly (because of the sharply vanishing exponential terms multiplying the expansion in factorization (7). Thus we expect that combined with a dimensionality reduction technique, we can achieve very competitive solutions.

In this paper we explore the application of the IFGT accelerated RLSC to certain standard problems that have already been solved by the other techniques. While dimensionality reduction would be desirable, here we do not perform such a reduction for fair comparison. We use small order expansions ($p = 1$ and $p = 2$) in the IFGT and run the iterative solver.

In the first experiment, we compared the performance of the IFGT on approximating the sums (4) with the Nyström method [19]. The experiments were carried out on a Pentium 4 1.4GHz PC with 512MB memory. We generate $N$ source points and $N$ target points in 100 dimensional unit hypercubes using a uniform distribution. The weights on the source points are generated using a uniform distribution in the interval $[0, 1]$. We directly evaluate the sums (4) as the ground truth, where $\sigma^2 = (0.5)d$ and $d$ is the dimensionality of the data. Then we estimate it using the improved fast Gauss transform and Nyström method. To compare the results, we use the maximum relative error to measure the precision of the approximations. Given a precision of $0.5\%$, we use the error bound (11) to find the parameters of the IFGT, and use a trial and error method to find the parameter of the Nyström method. Then we vary the number of points, $N$, from $500$ to $5000$ and plot the time against $N$ in Figure 2 (a). The results show the IFGT is much faster than the Nyström method. We also fix the number of points to $N = 1000$ and vary the size of centers (or random subset) $k$ from $10$ to $1000$ and plot the results in Figure 2 (b). The results show that the errors of the IFGT are not sensitive to the number of the centers, which means we can use very a small number of centers to achieve a good approximation. The accuracy of the Nyström

method catches up at large $k$, where the direct evaluation may be even faster. The intuition is that the use of expansions improves the accuracy of the approximation and relaxes the requirement of the centers.

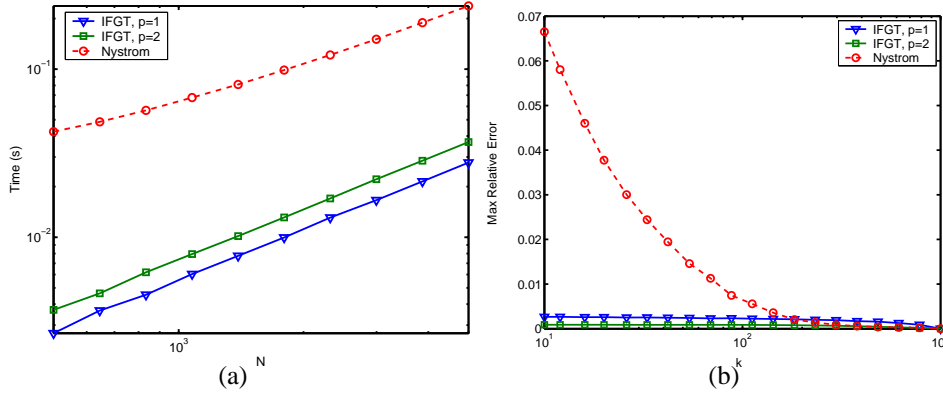

**Figure 2:** Performance comparison between the approximation methods. (a) Running time against $N$ and (b) maximum relative error against $k$ for fixed $N = 1000$ in 100 dimensions.

**Table 1:** Ten-fold training and testing accuracy in percentage and training time in seconds using the four classifiers on the five UCI datasets. Same value of $\hat{\sigma} = (0.5)d$ is used in all the classifiers. A rectangular kernel matrix with random subset size of 20% of $N$ was used in PSVM on *Galaxy Dim* and *Mushroom* datasets.

| Dataset | RLSC+FGT | RLSC | Nyström | PSVM |
|---|---|---|---|---|
| Size × Dimension | %, %, $s$ | %, %, $s$ | %, %, $s$ | %, %, $s$ |
| Ionosphere | 94.8400 | 97.7209 | 91.8656 | 95.1250 |
| 251 × 34 | 91.7302 | 90.6032 | 88.8889 | 94.0079 |
|  | 0.3711 | 1.1673 | 0.4096 | 0.8862 |
| BUPA Liver | 79.6789 | 81.7318 | 76.7488 | 75.8134 |
| 345 × 6 | 71.0336 | 67.8403 | 69.2857 | 71.4874 |
|  | 0.1279 | 0.4833 | 0.1475 | 0.3468 |
| Tic-Tac-Toe | 88.7263 | 88.6917 | 88.4945 | 92.9715 |
| 958 × 9 | 86.9507 | 85.4890 | 84.1272 | 87.2680 |
|  | 0.3476 | 2.9676 | 1.8326 | 3.9891 |
| Galaxy Dim | 93.2967 | 93.3206 | 93.7023 | 93.6705 |
| 4192 × 14 | 93.2014 | 93.2258 | 93.7020 | 93.5589 |
|  | 2.0972 | 78.3526 | 3.1081 | 44.5143 |
| Mushroom | 88.2556 | 87.9001 |  | 85.5955 |
| 8124 × 22 | 87.9615 | 87.6658 | failed | 85.4629 |
|  | 14.7422 | 341.7148 |  | 285.1126 |

In the second experiment, five datasets from the UCI repository are used to compare the performance of four different methods for classification: RLSC with the IFGT, RLSC with full kernel evaluation, RLSC with the Nyström method and the Proximal Support Vector Machines (PSVM) [4]. The Gaussian kernel is used for all these methods. We use the same value of $\sigma^2 = (0.5)d$ for a fair comparison. The ten-fold cross validation accuracy on training and testing and the training time are listed in Table 1. The RLSC with the IFGT is fastest among the four classifiers on all five datasets, while the training and testing accuracy is close to the accuracy of the RLSC with full kernel evaluation. The RLSC with the Nyström approximation is nearly as fast, but the accuracy is lower than the other methods. Worst of all, it is not always feasible to solve the linear systems, which results in the failure on the *Mushroom* dataset. The PSVM is accurate on the training and testing, but slow and memory demanding for large datasets, even with subset reduction.

# 5 Conclusions and Discussion

We presented an improved fast Gauss transform to speed up kernel machines with Gaussian kernel to linear order. The simulations and the classification experiments show that the algorithm is in general faster and more accurate than other matrix approximation methods. At present, we do not consider the reduction from the support vector set or dimensionality reduction. The combination of the improved fast Gauss transform with these techniques should bring even more reduction in computation. Another improvement to the algorithm is an automatic procedure to tune the parameters. A possible solution could be running a series of testing problems and tuning the parameters accordingly. If the bandwidth is very small compared with the data range, the nearest neighbor searching algorithms could be a better solution to these problems.

**Acknowledgments**

We would like to thank Dr. Nail Gumerov for many discussions. We also gratefully acknowledge support of NSF awards 9987944, 0086075 and 0219681.

# References

[1] M. Bern and D. Eppstein. Approximation algorithms for geometric problems. In D. Hochbaum, editor, *Approximation Algorithms for NP-Hard Problems*, chapter 8, pages 296–345. PWS Publishing Company, Boston, 1997.

[2] T. Feder and D. Greene. Optimal algorithms for approximate clustering. In *Proc. 20th ACM Symp. Theory of computing*, pages 434–444, Chicago, Illinois, 1988.

[3] S. Fine and K. Scheinberg. Efficient SVM training using low-rank kernel representations. *Journal of Machine Learning Research*, 2:243–264, Dec. 2001.

[4] G. Fung and O. L. Mangasarian. Proximal support vector machine classifiers. In *Proceedings KDD-2001: Knowledge Discovery and Data Mining*, pages 77–86, San Francisco, CA, 2001.

[5] F. Girosi, M. Jones, and T. Poggio. Regularization theory and neural networks architectures. *Neural Computation*, 7(2):219–269, 1995.

[6] T. Gonzalez. Clustering to minimize the maximum intercluster distance. *Theoretical Computer Science*, 38:293–306, 1985.

[7] L. Greengard and V. Rokhlin. A fast algorithm for particle simulations. *J. Comput. Phys.*, 73(2):325–348, 1987.

[8] L. Greengard and J. Strain. The fast Gauss transform. *SIAM J. Sci. Statist. Comput.*, 12(1):79–94, 1991.

[9] Y.-J. Lee and O. Mangasarian. RSVM: Reduced support vector machines. In *First SIAM International Conference on Data Mining*, Chicago, 2001.

[10] T. Poggio and S. Smale. The mathematics of learning: Dealing with data. *Notices of the American Mathematical Society (AMS)*, 50(5):537–544, 2003.

[11] R. Rifkin. *Everything Old Is New Again: A Fresh Look at Historical Approaches in Machine Learning*. PhD thesis, MIT, Cambridge, MA, 2002.

[12] A. Smola and P. Bartlett. Sparse greedy gaussian process regression. In *Advances in Neural Information Processing Systems*, pages 619–625. MIT Press, 2001.

[13] A. Smola and B. Schölkopf. Sparse greedy matrix approximation for machine learning. In *Proc. Int'l Conf. Machine Learning*, pages 911–918. Morgan Kaufmann, 2000.

[14] X. Sun and N. P. Pitsianis. A matrix version of the fast multipole method. *SIAM Review*, 43(2):289–300, 2001.

[15] J. A. K. Suykens and J. Vandewalle. Least squares support vector machine classifiers. *Neural Processing Letters*, 9(3):293–300, 1999.

[16] V. Vapnik. *The Nature of Statistical Learning Theory*. Springer, New York, 1995.

[17] G. Wahba. *Spline Models for Observational Data*. SIAM, Philadelphia, PA, 1990.

[18] C. K. Williams and D. Barber. Bayesian classification with gaussian processes. *IEEE Trans. Pattern Anal. Mach. Intell.*, 20(12):1342–1351, Dec. 1998.

[19] C. K. I. Williams and M. Seeger. Using the Nyström method to speed up kernel machines. In *Advances in Neural Information Processing Systems*, pages 682–688. MIT Press, 2001.

[20] C. Yang, R. Duraiswami, N. Gumerov, and L. Davis. Improved fast Gauss transform and efficient kernel density estimation. In *Proc. ICCV 2003*, pages 464–471, 2003.

[21] C. Yang, R. Duraiswami, and N. A. Gumerov. Improved fast gauss transform. Technical Report CS-TR-4495, UMIACS, Univ. of Maryland, College Park, 2003.
